# NEURONAL MAPS FOR SENSORY-MOTOR CONTROL IN THE BARN OWL

C.D. Spence, J.C. Pearson, J.J. Gelfand, and R.M. Peterson
David Sarnoff Research Center
Subsidiary of SRI International
CN5300
Princeton, New Jersey 08543-5300

W.E. Sullivan
Department of Biology
Princeton University
Princeton, New Jersey 08544

## ABSTRACT

The barn owl has fused visual/auditory/motor representations of space in its midbrain which are used to orient the head so that visual or auditory stimuli are centered in the visual field of view. We present models and computer simulations of these structures which address various problems, including the construction of a map of space from auditory sensory information, and the problem of driving the motor system from these maps. We compare the results with biological data.

## INTRODUCTION

Many neural network models have little resemblance to real neural structures, partly because the brain's higher functions, which they attempt to imitate, are not yet experimentally accessible. Nevertheless, some neural-net researchers are finding that the accessible structures are interesting, and that their functions are potentially useful. Our group is modeling a part of the barn owl's nervous system which orients the head to visual and auditory stimuli.

The barn owl's brain stem and midbrain contains a system that locates visual and auditory stimuli in space. The system constructs an auditory map of spatial direction from the non-spatial information in the output of the two cochlea. This map is in the external nucleus of the inferior colliculus, or ICx [Knudsen and Konishi, 1978]. The ICx, along with the visual system, projects to the optic tectum, producing a fused visual and auditory map of space [Knudsen and Knudsen, 1983]. The map in the tectum is the source of target position used by the motor system for orienting the head.

In the last fifteen years, biologists have determined all of the structures in the system which produces the auditory map of space in the ICx. This system provides sev-

eral examples of neuronal maps, regions of tissue in which the response properties of neurons vary continuously with position in the map. (For reviews, see Knudsen, 1984; Knudsen, du Lac, and Esterly, 1987; and Konishi, 1986.) Unfortunately, the motor system and the projections from the tectum are not well known, but experimental study of them has recently begun [Masino and Knudsen, 1988]. We should eventually be able to model a complete system, from sensory input to motor output.

In this paper we present several models of different parts of the head orientation system. Fig. 1 is an overview of the structures we'll discuss. In the second section of this paper we discuss models for the construction of the auditory space map in the ICx. In the third section we discuss how the optic tectum might drive the motor system.

## CONSTRUCTION OF AN AUDITORY MAP OF SPACE

The barn owl uses two binaural cues to locate objects in space: azimuth is derived from inter-aural time or phase delay (not onset time difference), while elevation is derived from inter-aural intensity difference (due to vertical asymmetries in sensitiv-

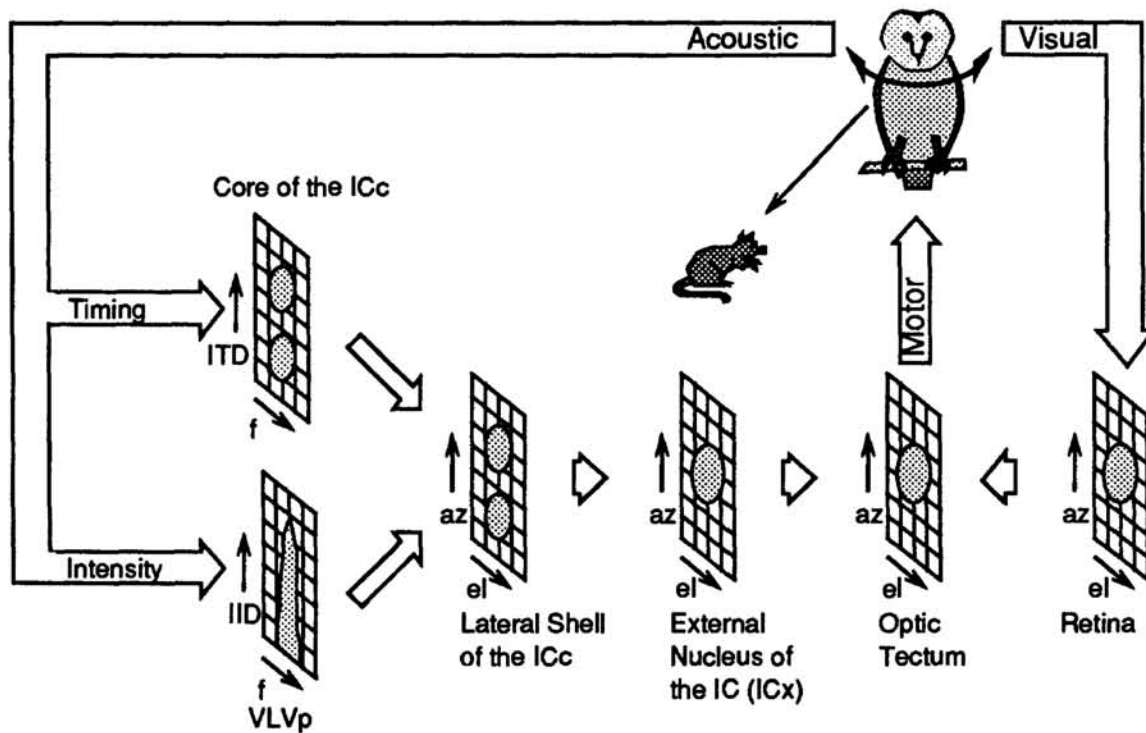

**Figure 1.** Overview of the neuronal system for target localization in the barn owl (head orients towards potential targets for closer scrutiny). The illustration focuses on the functional representations of the neuronal computation, and does not show all of the relevant connections. The grids represent the centrally synthesized neuronal maps and the patterns within them indicate possible patterns of neuronal activation in response to acoustic stimuli.

ity). Corresponding to these two cues are two separate processing streams which produce maps of the binaural cues, which are shown in Figs. 2-5. The information on these maps must be merged in order to construct the map of space in the ICx.

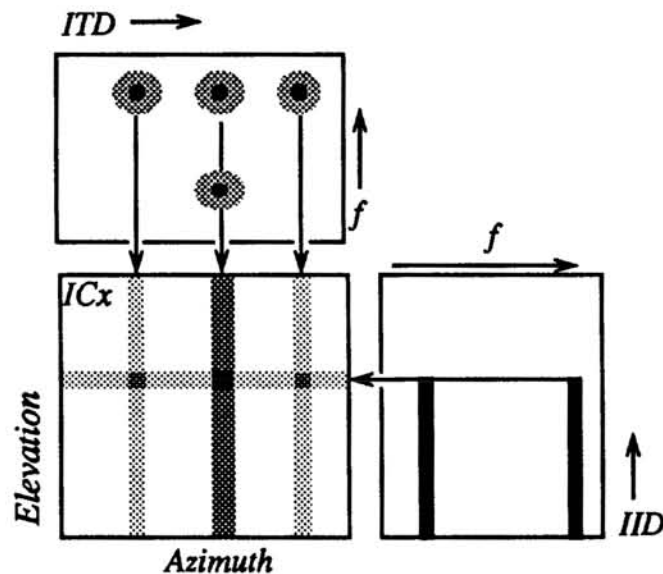

**Figure 2.** Standard Model for the construction of an auditory map of space from maps of the binaural cues. Shading represents activity level. IID is Inter-aural Intensity Difference, ITD is Inter-aural Time Delay.

A simple model for combining the two maps is shown in Fig. 2. It has not been described explicitly in the literature, but it has been hinted at [Knudsen, et al, 1987]. For this reason we have called it the standard model. Here all of the neurons representing a given time delay or azimuth in the ITD vs. frequency map project to all of the neurons representing that azimuth in the space map. Thus a stimulus with a certain ITD would excite a strip of cells representing the associated azimuth and all elevations. Similarly, all of the neurons representing a given intensity difference or elevation in the IID vs. frequency map project to all of the neurons representing that elevation in the space map. (The map of IID vs. frequency is constructed in the nucleus ventralis lemnisci lateralis pars posterior, or VLVp. VLVp neurons are said to be sensitive to intensity difference, that is they fire if the intensity difference is great enough. Neurons in the VLVp are spatially organized by their intensity difference threshold [Manley, et al, 1988]. Thus, intensity difference has a bar-chart-like representation, and our model needs some mechanism to pick out the ends of the bars.) Only the neurons at the intersection of the two strips will fire if lateral inhibition allows only those neurons receiving the most stimulation to fire. In the third section we will present a model for connections of inhibitory inter-neurons which can be applied to this model.

Part of the motivation for the standard model is the problem with phase ghosts. Phase ghosts occur when the barn owl's nervous system incorrectly associates the wave fronts arriving at the two ears at high frequency. In this case, neurons in the map of ITD vs. frequency become active at locations representing a time delay which

differs from the true time delay by an integer multiple of the period of the sound. Because the period varies inversely with the frequency, these phase ghosts will have apparent time delays that vary with frequency. Thus, for stimuli that are not pure tones, if the barn owl can compare the activities in the map at different frequencies, it can eliminate the ghosts. The standard model does this (Fig. 2). In the ITD vs. frequency map there are more neurons firing at the position of the true ITD than at the ghost positions, so space map neurons representing the true position will receive the most stimulation. Only those neurons representing the true position will fire because of the lateral inhibition.

There is another kind of ghost which we call the multiple-source ghost (Fig. 3). If two sounds occur simultaneously, then space map neurons representing the time delay of one source and the intensity difference of the other will receive a large amount of stimulation. Lateral inhibition may suppress these ghosts, but if so, the owl should only be able to locate one source at a time. In addition, the true sources might be suppressed. The barn owl may actually suffer from this problem, although it seems unlikely if the owl has to function in a noisy environment. The relevant behavioral experiments have not yet been done.

Experimental evidence does not support the standard model. The ICx receives most of its input from the lateral shell of the central nucleus of the inferior colliculus

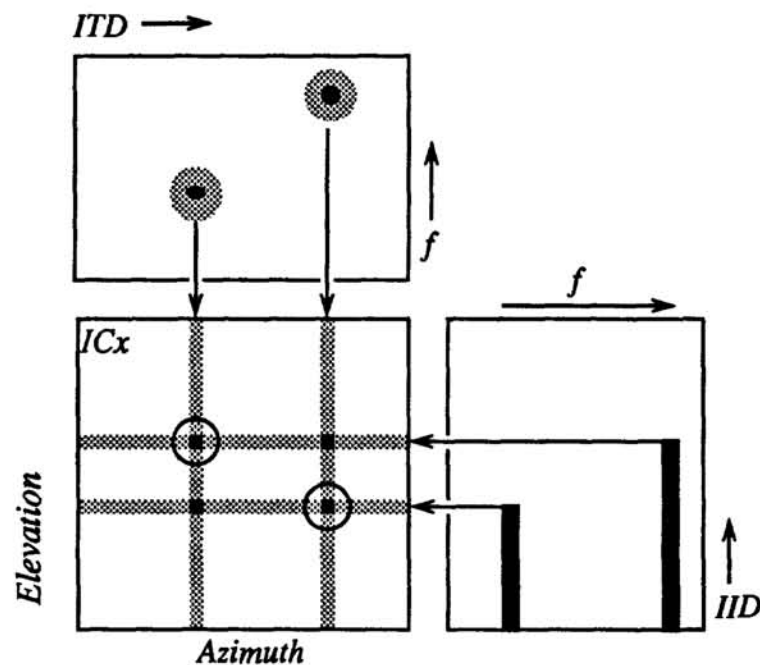

**Figure 3.** Multiple-source ghosts in the standard model for the construction of the auditory space map. For clarity, only two pure tone stimuli are represented, and their frequencies and locations are such that the "phase ghost" problem is not a factor. The black squares represent regions of cells that are above threshold. The circled regions are those that are firing in response to the ITD of one stimulus and the IID of another. These regions correspond to phantom targets.

(lateral shell of the ICc) [Knudsen, 1983]. Neurons in the lateral shell are tuned to frequency and time delay, and these parameters are mapped [Wagner, Takahashi, and Konishi, 1987]. However, they are also affected by intensity difference [Knudsen and Konishi, 1978, I. Fujita, private communication]. Thus the lateral shell does not fit the picture of the input to the ICx in the standard model, rather it is some intermediate stage in the processing.

We have a model, called the lateral shell model, which does not suffer from multiple-source ghosts (Fig. 4). In this model, the lateral shell of the ICc is a three-dimensional map of frequency vs. intensity difference vs. time delay. A neuron in the map of time delay vs. frequency in the ICc core projects to all of the neurons in

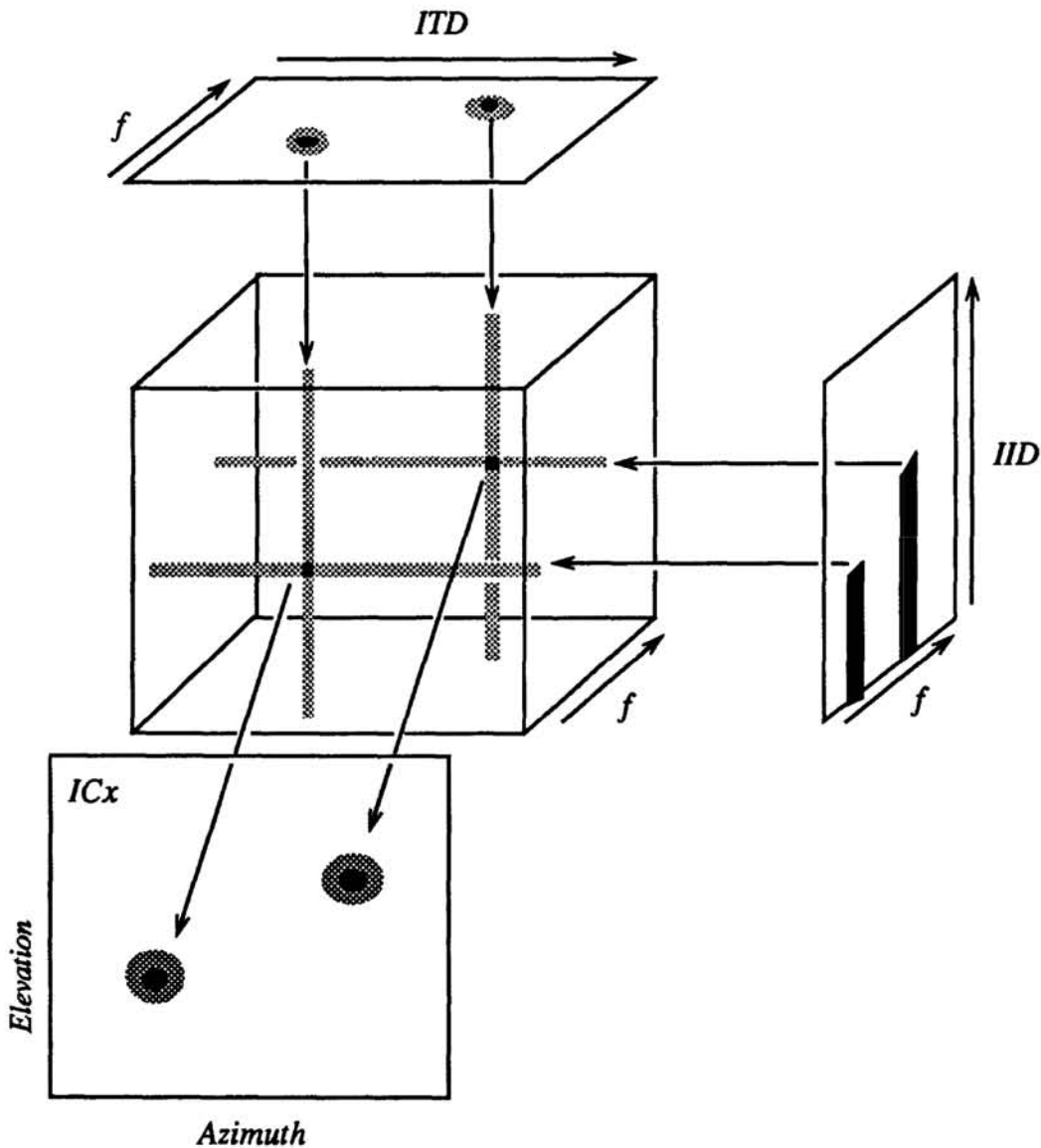

**Figure 4.** Lateral shell model for the construction of the auditory map of space in the ICx. *f*: frequency, *ITD*: inter-aural time delay, *IID*: inter-aural intensity difference.

the three-dimensional map which represent the same time delay and frequency. As in the standard model, a strip of neurons is stimulated, but now the frequency tuning is preserved. The map of intensity difference vs. frequency in the nucleus ventralis lemnisci lateralis pars posterior (VLVp) [Manley, et al, 1988] projects to the three-dimensional map in a similar fashion. Lateral inhibition picks out the regions of intersection of the strips. Neurons in the space map in the ICx receive input from the strip of neurons in the three-dimensional map which represent the appropriate time delay and intensity difference, or equivalently azimuth and elevation. Phase ghosts will be present in the three-dimensional map, but in the ICx lateral inhibition will suppress them.

Multiple-source ghosts are eliminated in the lateral shell model because the sources are essentially tagged by their frequency spectra. If two sources with no common frequencies are present, there are no neurons which represent the time delay of one source, the intensity difference of another, and a frequency common to both. In the more likely case in which some frequencies are common to both sources, there will be fewer neurons firing at the ghost positions than at the real positions, so again lateral inhibition in the ICx can suppress firing at the ghost positions, exactly as it suppresses the phase ghosts. The fact that intensity and time delay information is combined before frequency tuning is lost in the ICx suggests that the owl handles multiple-source ghosts by frequency tagging. A three dimensional map is not essential, but it is conceptually simple.

Before ending this section, we should mention that others have independently thought of this model. In particular, M. Konishi and co-workers have looked for a spatial organization or mapping of intensity response properties in the lateral shell, but they have not found it. They also have said that they can't yet rule it out [M. Konishi, I. Fujita, private communication].

## DRIVING THE MOTOR SYSTEM FROM MAPS

As mentioned before, all of the parts of the auditory system in the brain stem and midbrain are known, up to the optic tectum. The optic tectum has a map of spatial direction which is common to both the visual and auditory systems. In addition, it drives the motor system, so if the tectum is stimulated at a point, the barn owl's head will move to face a certain direction. The new orientation is mapped in register with the auditory/visual map of spatial direction, e.g., stimulating a location which represents a stimulus eight degrees to the right of the current orientation of the head will cause the head to turn eight degrees to the right.

Little is known about the projections of the tectum, although work has started [Masino and Knudsen, 1988]. There was one earlier experiment. Two electrodes were placed in one of the tecta at positions representing sensory stimuli eight degrees and sixty degrees toward the side of the head opposite to this tectum. When either position was stimulated by itself, the alert owl moved its head as expected, by eight or

sixty degrees. When both were stimulated together, the head moved about forty degrees [du Lac and Knudsen, 1987].

The averaging of activity in the tectum is easy to explain. In some motor models it should be produced naturally by the activation of an agonist-antagonist muscle pair (see, for example, Grossberg and Kuperstein, 1986). In the presence of two stimuli, the tension in each muscle is the sum of the tensions it would have for either stimulus alone, so the equilibrium position should be about the average position.

We have a different model, which produces a map of the average position. The connection strengths from tectal cells to an averaging map cell decrease quadratically with the difference in represented direction. A quadratic distribution of stimulation is very broad, however, so lateral inhibition is required to make the active region fairly narrow.

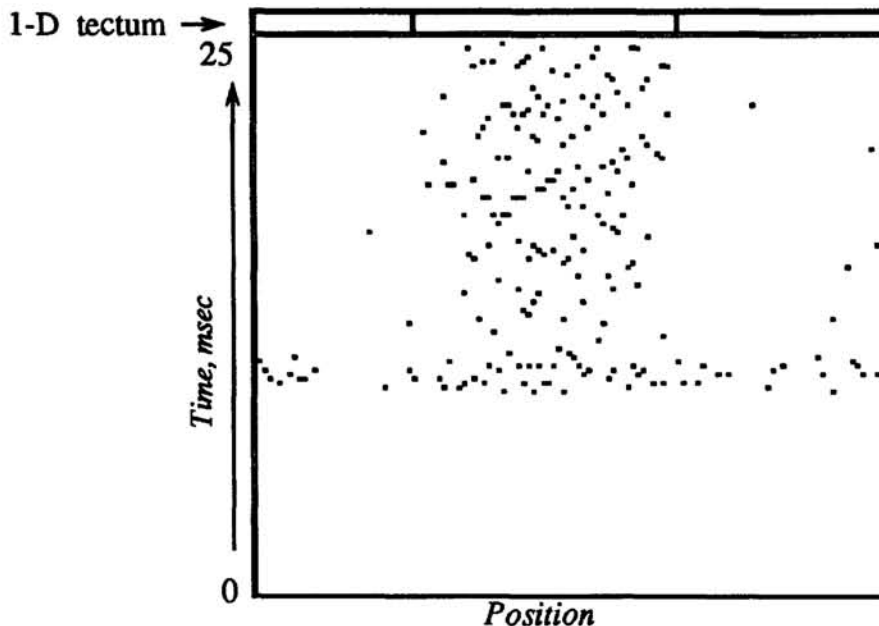

**Figure 5.** Averaging map simulation. The upper thin rectangle is a one-dimensional version of the space map in the optic tectum. The two marks represent the position of stimulating electrodes that are simultaneously active. The lower rectangle is the averaging map, with position represented horizontally and time increasing vertically. The squares represent cell firings. Note that the activity quickly becomes centered at the average position of the two active positions in the tectum.

We have simulated a one-dimensional version of this model, with 128 cells in the tectum, and in the averaging map, 128 excitatory and 128 inhibitory cells. The excitatory cells and the inhibitory cells both receive the same quadratically weighted input from the tectum. Each inhibitory cell inhibits all of the other cells in the averaging map, both excitatory and inhibitory, except for those in a small local neighborhood. (The weights are actually proportional to one minus a gaussian with a maximum of one.) An excitatory cell receives exactly the same input as the inhibitory

cell at the same location, so their voltages are the same. Because of this we only show the excitatory cells in Fig. 5. This figure shows cell position horizontally and time increasing vertically. The black squares are plotted at the time a given cell fires.

We are interested in whether an architecture like this is biologically plausible. For this reason we have tried to be fairly realistic in our model. The cells obey a membrane equation:

$$C \frac{dv}{dt} = -g_l (v - v_l) - g_e (v - v_e) - g_i (v - v_i)$$

in which $C$ is the capacitance, the $g$'s are conductances, $l$ refers to leakage quantities, $e$ refers to excitatory quantities, and $i$ refers to inhibitory quantities. The output of a cell is not a real number, but spikes that occur randomly at an average rate that is a monotonic function of the voltage. We used the usual sigmoidal function in this simulation, although the membrane equation automatically limits the voltage and hence the firing rate. A cell that spikes on a given time step affects other cells by affecting the appropriate conductance. To get the effect of a post synaptic potential, the conductances obey the equation of a damped harmonic oscillator:

$$\frac{d^2g}{dt^2} = -\gamma \frac{dg}{dt} - \omega^{-2} g$$

When a spike from an excitatory cell arrives, we increment the time derivative of $g$ by some amount. If the oscillator is overdamped or critically damped, the conductance goes up for a time and then decreases, approaching zero exponentially.

We are not suggesting that a damped harmonic oscillator exists in the membranes of neurons, but it efficiently models the dynamics of synaptic transmission. The equations for the conductances also have the nice property that the effects of multiple spikes at different times add.

With values for the cell parameters that agree well with experimental data, it takes about twenty milliseconds for the simulated map to settle into a fairly steady state, which is a reasonable time for the function of this map. Also, there was no need to fine tune the parameters; within a fairly wide range the effect of changing them is to change the width of the region of activity.

We tried another architecture for the inhibitory interneurons, in which they received their input from the excitatory neurons and did not inhibit other inhibitory neurons. The voltages in this architecture oscillated for a very long time, without picking out a maximum. The architecture we are now using is apparently superior. Since it is quick to pick out a maximum of a broad distribution of stimulation, it should work very well in other models requiring lateral inhibition, such as the lateral shell model discussed earlier.

# CONCLUSION

We have presented models for two parts of the barn owl's visual/auditory localization and head orientation system. These models make experimentally testable predictions, and suggest architectures for artificial systems. One model constructs a map of stimulus position from maps of inter-aural intensity and timing differences. This model solves potential problems with ghosts, i.e., the representation of false sources in the presence of certain kinds of real sources.

Another model computes the average value of a quantity represented on a neural map when the activity on the map has a complex distribution. This model explains recent physiological experiments. A simulation with fairly realistic model neurons has shown that a biological structure could perform this function in this way.

A common feature of these models is the use of neuronal maps. We have only mentioned a few of the maps in the barn owl, and they are extremely common in other nervous systems. We think this architecture shows great promise for applications in artificial processing systems.

## Acknowledgments

This work was supported by internal funds of the David Sarnoff Research Center.

## References

Grossberg, S. and M. Kuperstein (1986) *Neural dynamics of adaptive motor control*, North-Holland.

Knudsen, E.I. (1983) J. Comp. Neurology, 218:174-186.

Knudsen, E.I. (1984) in *Dynamical Aspects of Neocortical Function*, G.M. Edelman, W.E. Gall, and W.M. Cowan, editors, Wiley, New York.

Knudsen, E.I. and P.F. Knudsen (1983) J. Comp. Neurology, 218:187-196.

Knudsen, E.I. and M. Konishi (1978) J. Neurophys., 41:870-884.

Knudsen, E.I., S. du Lac, and S. Esterly (1987) Ann. Rev. Neurosci., 10:41-56.

Konishi, M. (1986) Trends in Neuroscience, April.

du Lac, S. and E.I. Knudsen (1987) Soc. for Neurosci. Abstr., 112.10.

Manley, G.A., A.C. Koeppl, and M. Konishi (1988) J. Neurosci. 8:2665-2676

Masino, T., and E.I. Knudsen (1988) Soc. for Neurosci. Abstr., 496.16.

Wagner, H., T. Takahashi, and M. Konishi (1987) J. Neurosci. 10:3105-3116